# Learning Graphical Models with Mercer Kernels

**Francis R. Bach**
Division of Computer Science
University of California
Berkeley, CA 94720
*fbach@cs.berkeley.edu*

**Michael I. Jordan**
Computer Science and Statistics
University of California
Berkeley, CA 94720
*jordan@cs.berkeley.edu*

## Abstract

We present a class of algorithms for learning the structure of graphical models from data. The algorithms are based on a measure known as the *kernel generalized variance (KGV)*, which essentially allows us to treat all variables on an equal footing as Gaussians in a feature space obtained from Mercer kernels. Thus we are able to learn hybrid graphs involving discrete and continuous variables of arbitrary type. We explore the computational properties of our approach, showing how to use the kernel trick to compute the relevant statistics in linear time. We illustrate our framework with experiments involving discrete and continuous data.

## 1   Introduction

Graphical models are a compact and efficient way of representing a joint probability distribution of a set of variables. In recent years, there has been a growing interest in learning the structure of graphical models directly from data, either in the directed case [1, 2, 3, 4] or the undirected case [5]. Current algorithms deal reasonably well with models involving discrete variables or Gaussian variables having only limited interaction with discrete neighbors. However, applications to general hybrid graphs and to domains with general continuous variables are few, and are generally based on discretization.

In this paper, we present a general framework that can be applied to any type of variable. We make use of a relationship between kernel-based measures of "generalized variance" in a feature space, and quantities such as mutual information and pairwise independence in the input space. In particular, suppose that each variable $x_i$ in our domain is mapped into a high-dimensional space $\mathcal{F}_i$ via a map $\Phi_i$. Let $\phi_i = \Phi_i(x_i)$ and consider the set of random variables $\{\phi_i\}$ in feature space. Suppose that we compute the mean and covariance matrix of these variables and consider a set of Gaussian variables, $\{\phi_i^G\}$, that have the same mean and covariance. We showed in [6] that a canonical correlation analysis of $\{\phi_i^G\}$ yields a measure, known as "kernel generalized variance," that characterizes pairwise independence among the original variables $\{x_i\}$, and is closely related to the mutual information among the original variables. This link led to a new set of algorithms for independent component analysis. In the current paper we pursue this idea in a different direction, considering the use of the kernel generalized variance as a surrogate for the mutual information in model selection problems. Effectively, we map data into a feature space via a set of Mercer kernels, with different kernels for different data types, and treat all data on an equal footing

as Gaussian in feature space.

We briefly review the structure-learning problem in Section 2, and in Section 4 and Section 5 we show how classical approaches to the problem, based on MDL/BIC and conditional independence tests, can be extended to our kernel-based approach. In Section 3 we show that by making use of the "kernel trick" we are able to compute the sample covariance matrix in feature space in linear time in the number of samples. Section 6 presents experimental results.

## 2   Learning graphical models

Structure learning algorithms generally use one of two equivalent interpretations of graphical models [7]: the compact factorization of the joint probability distribution function leads to local search algorithms while conditional independence relationships suggest methods based on conditional independence tests.

**Local search.** In this approach, structure learning is explicitly cast as a model selection problem. For directed graphical models, in the MDL/BIC setting of [2], the likelihood is penalized by a model selection term that is equal to $\frac{1}{2}\log N$ times the number of parameters necessary to encode the local distributions. The likelihood term can be decomposed and expressed as follows: $J_{ML} = \sum_i J_{ML}(i, \pi_i)$, with $J_{ML}(i, \pi_i) = -NI(x_i, x_{\pi_i})$, where $\pi_i$ is the set of parents of node $i$ in the graph to be scored and $I(x_i, x_{\pi_i})$ is the empirical mutual information between the variable $x_i$ and the vector $x_{\pi_i}$. These mutual information terms and the number of parameters for each local conditional distributions are easily computable in discrete models, as well as in Gaussian models. Alternatively, in a full Bayesian framework, under assumptions about parameter independence, parameter modularity, and prior distributions (Dirichlet for discrete networks, inverse Wishart for Gaussian networks), the log-posterior probability of a graph given the data can be decomposed in a similar way [1, 3].

Given that our approach is based on the assumption of Gaussianity in feature space, we could base our development on either the MDL/BIC approach or the full Bayesian approach. In this paper, we extend the MDL/BIC approach, as detailed in Section 4.

**Conditional independence tests.** In this approach, conditional independence tests are performed to constrain the structure of possible graphs. For undirected models, going from the graph to the set of conditional independences is relatively easy: there is an edge between $x_i$ and $x_j$ if and only if $x_i$ and $x_j$ are independent given all other variables [7]. In Section 5, we show how our approach could be used to perform independence tests and learn an undirected graphical model. We also show how this approach can be used to prune the search space for the local search of a directed model.

## 3   Gaussians in feature space

In this section, we introduce our Gaussianity assumption and show how to approximate the mutual information, as required for the structure learning algorithms.

### 3.1   Mercer Kernels

A *Mercer kernel* on a space $\mathcal{X}$ is a function $k(x, y)$ from $\mathcal{X}^2$ to $\mathbb{R}$ such that for any set of points $\{x^1, \ldots, x^N\}$ in $\mathcal{X}$, the $N \times N$ matrix $K$, defined by $K_{ij} = k(x_i, x_j)$, is positive semidefinite. The matrix $K$ is usually referred to as the *Gram matrix* of the points $\{x^i\}$. Given a Mercer kernel $k(x, y)$, it is possible to find a space $\mathcal{F}$ and a map $\Phi$ from $\mathcal{X}$ to $\mathcal{F}$, such that $k(x, y)$ is the dot product in $\mathcal{F}$ between $\Phi(x)$ and $\Phi(y)$ (see, e.g., [8]). The space $\mathcal{F}$ is usually referred to as the *feature space* and the map $\Phi$ as the *feature map*. We will use

the notation $f^\top g$ to denote the dot product of $f$ and $g$ in feature space $\mathcal{F}$. We also use the notation $f^\top$ to denote the representative of $f$ in the dual space of $\mathcal{F}$.

For a discrete variable which takes values in $\{1, \ldots, d\}$, we use the *trivial kernel* $k(x, y) = \delta_{x=y}$, which corresponds to a feature space of dimension $d$. The feature map is $\Phi(x) = (\delta_{x=1}, \ldots, \delta_{x=d})$. Note that this mapping corresponds to the usual embedding of a multinomial variable of order $d$ in the vector space $\mathbb{R}^d$.

For continuous variables, we use the Gaussian kernel $k(x, y) = e^{-(x-y)^2/2\sigma^2}$. The feature space has infinite dimension, but as we will show, the data only occupy a small linear manifold and this linear subspace can be determined adaptively in linear time. Note that an alternative is to use the kernel $k(x, y) = xy$, which corresponds to simply modeling the data as Gaussian in input space.

### 3.2 Notation

Let $x_1, \ldots, x_m$ be $m$ random variables with values in spaces $\mathcal{X}_1, \ldots, \mathcal{X}_m$. Let us assign a Mercer kernel $k_i$ to each of the input spaces $\mathcal{X}_i$, with feature space $\mathcal{F}_i$ and feature map $\Phi_i$. The random vector of feature images $\phi = (\phi_1, \ldots, \phi_m) \triangleq (\Phi_1(x_1), \ldots, \Phi_m(x_m))$ has a covariance matrix $C$ defined by blocks, with block $C_{ij}$ being the covariance matrix between $\phi_i = \Phi_i(x_i)$ and $\phi_j = \Phi_j(x_j)$. Let $\phi^G = (\phi_1^G, \ldots, \phi_m^G)$ denote a jointly Gaussian vector with the same mean and covariance as $\phi = (\phi_1, \ldots, \phi_m)$. The vector $\phi^G$ will be used as the random vector on which the learning of graphical model structure is based.

Note that the sufficient statistics for this vector are $\{\Phi_i(x_i), \Phi_i(x_i)\Phi_j(x_j)^\top\}$, and are inherently pairwise. No dependency involving strictly more than two variables is modeled explicitly, which makes our scoring metric easy to compute. In Section 6, we present empirical evidence that good models can be learned using only pairwise information.

### 3.3 Computing sample covariances using kernel trick

We are given a random sample $\{x^1, \ldots, x^N\}$ of elements of $\mathcal{X}_1 \times \ldots \times \mathcal{X}_m$. By mapping into the feature spaces, we define $Nm$ elements $\phi_i^k = \Phi_i(x_i^k)$. We assume that for each $i$ the data in feature space $\{\phi_i^1, \ldots, \phi_i^N\}$ have been centered, i.e., $\sum_{k=1}^N \phi_i^k = 0$. The sample covariance matrix $\hat{C}_{ij}$ is then equal to $\hat{C}_{ij} = \frac{1}{N} \sum_{k=1}^N \phi_i^k (\phi_j^k)^\top$. Note that a Gaussian with covariance matrix $\hat{C}$ has zero variance along directions that are orthogonal to the images of the data. Consequently, in order to compute the mutual information, we only need to compute the covariance matrix of the projection of $\phi$ onto the linear span of the data, that is, for all $i, j, s, t$:

$$(\phi_i^s)^\top \hat{C}_{ij}\, \phi_j^t = \frac{1}{N} \sum_{k=1}^N (\phi_i^s)^\top \phi_i^k\, (\phi_j^k)^\top \phi_j^t = \frac{1}{N} \sum_{k=1}^N (K_i)_{sk}(K_j)_{tk} = \frac{1}{N}\delta_s^\top K_i K_j \delta_t, \quad (1)$$

where $\delta_s$ denotes the $N \times 1$ vectors with only zeros except at position $s$, and $K_i$ is the Gram matrix of the centered points, the so-called centered Gram matrix of the $i$-th component, defined from the Gram matrix $L_i$ of the original (non-centered) points as $K_i = (I - \frac{1}{N}\mathbf{1})\, L_i\, (I - \frac{1}{N}\mathbf{1})$, where $\mathbf{1}$ is a $N \times N$ matrix composed of ones [8]. From Eq. (1), we see that the sample covariance matrix of $\phi$ in the "data basis" has blocks $\frac{1}{N}K_i K_j$.

### 3.4 Regularization

When the feature space has infinite dimension (as in the case of a Gaussian kernel on $\mathbb{R}$), then the covariance we are implicitly fitting with a kernel method has an infinite number of parameters. In order to avoid overfitting and control the capacity of our models, we

regularize by smoothing the Gaussian $\phi^G$ by another Gaussian with small variance (for an alternative interpretation and further details, see [6]). Let $\kappa$ be a small constant. We add to $\phi^G$ an isotropic Gaussian with covariance $2\kappa I$ in an orthonormal basis. In the data basis, the covariance of this Gaussian is exactly the block diagonal matrix with blocks $2\kappa K_i$. Consequently, our regularized Gaussian covariance $\tilde{C}$ has blocks $\tilde{C}_{ij} = \frac{1}{N} K_i K_j$ if $i \neq j$ and $\tilde{C}_{ii} = \frac{1}{N} K_i^2 + 2\kappa K_i$. Since $\kappa$ is a small constant, we can use $\tilde{C}_{ii} \approx \frac{1}{N}(K_i + N\kappa I)^2 = \frac{1}{N} K_i^2 + 2\kappa K_i + O(\kappa^2)$, which leads to a more compact correlation matrix $R$, with blocks $R_{ij} = R_i R_j$ for $i \neq j$, and $R_{ii} = I$, where $R_i = K_i(K_i + N\kappa I)^{-1}$. These cross-correlation matrices have exact dimension $N$, but since the eigenvalues of $K_i$ are softly thresholded to zero or one by the regularization, the effective dimension is $d_i = \text{tr}(K_i(K_i + N\kappa I)^{-1})$. This dimensionality $d_i$ will be used as the dimension of our Gaussian variables for the MDL/BIC criterion, in Section 4.

### 3.5 Efficient implementation

Direct manipulation of $N \times N$ matrices would lead to algorithms that scale as $O(N^3)$. Gram matrices, however, are known to be well approximated by matrices of low rank $M$. The approximation is exact when the feature space has finite dimension $d$ (e.g., with discrete kernels), and $M$ can be chosen less than $d$. In the case of continuous data with the Gaussian kernel, we have shown that $M$ can be chosen to be upper bounded by a constant independent of $N$ [6]. Finding a low-rank decomposition can thus be done through incomplete Cholesky decomposition in linear time in $N$ (for a detailed treatment of this issue, see [6]).

Using the incomplete Cholesky decomposition, for each matrix $K_i$ we obtain the factorization $K_i \approx G_i G_i^\top$, where $G_i$ is an $N \times M_i$ matrix with rank $M_i$, where $M_i \ll N$. We perform a singular value decomposition of $G_i$ to obtain an $N \times M_i$ matrix $U_i$ with orthogonal columns (i.e., such that $U_i^\top U_i = I$), and an $M_i \times M_i$ diagonal matrix $\Lambda_i$ such that $K_i \approx G_i G_i^\top = U_i \Lambda_i U_i^\top$.

We have $R_i = (K_i + N\kappa I)^{-1} K_i = U_i D_i U_i^\top$, where where $D_i$ is the diagonal matrix obtained from the diagonal matrix $\Lambda_i$ by applying the function $\lambda \mapsto \lambda/(\lambda + N\kappa)$ to its elements. Thus $\phi_i^G$ has a correlation matrix with blocks $R_{ij} = D_i U_i^\top U_j D_j$ in the new basis defined by the columns of the matrices $U_i$, and these blocks will be used to compute the various mutual information terms.

### 3.6 KGV-mutual information

We now show how to compute the mutual information between $\phi_1^G, \ldots, \phi_m^G$, and we make a link with the mutual information of the original variables $x_1, \ldots, x_m$.

Let $y_1, \ldots, y_m$ be $m$ jointly Gaussian random vectors with covariance matrix $\Sigma$, defined in terms of blocks $\Sigma_{ij} = \text{cov}(y_i, y_j)$. The mutual information between the variables $y_1, \ldots, y_m$ is equal to (see, e.g., [9]):

$$I(y_1, \ldots, y_m) = -\frac{1}{2} \log \frac{|\Sigma|}{|\Sigma_{11}| \cdots |\Sigma_{mm}|}, \qquad (2)$$

where $|A|$ denotes the determinant of the matrix $A$. The ratio of determinants in this expression is usually referred to as the *generalized variance*, and is independent of the basis which is chosen to compute $\Sigma$.

Following Eq. (2), the mutual information between $\phi_1^G, \ldots, \phi_m^G$, which depends solely on the distribution of $x$, is equal to

$$I^K(x_1, \ldots, x_m) = -\frac{1}{2} \log \frac{|R|}{|R_{11}| \cdots |R_{mm}|}. \qquad (3)$$

We refer to this quantity as the *KGV-mutual information* (KGV stands for kernel generalized variance). It is always nonnegative and can also be defined for partitions of the variables into subsets, by simply partitioning the correlation matrix $R$ accordingly.

The KGV has an interesting relationship to the mutual information among the original variables, $x_1, \ldots, x_m$. In particular, as shown in [6], in the case of two discrete variables, the KGV is equal to the mutual information up to second order, when expanding around the manifold of distributions that factorize in the trivial graphical model (i.e. with independent components). Moreover, in the case of continuous variables, when the width $\sigma$ of the Gaussian kernel tend to zero, the KGV necessarily tends to a limit, and also provides a second-order expansion of the mutual information around independence.

This suggests that the KGV-mutual information might also provide a useful, computationally-tractable surrogate for the mutual information more generally, and in particular substitute for mutual information terms in objective functions for model selection, where even a rough approximation might suffice to rank models. In the remainder of the paper, we investigate this possibility empirically.

## 4 Structure learning using local search

In this approach, an objective function $J : G \mapsto \mathbb{R}$ measures the goodness of fit of the directed graphical model $G$, and is minimized. The MDL/BIC objective function for our Gaussian variables is easily derived. Let $\pi_i = \pi_i(G)$ be the set of parents of node $i$ in $G$. We have $J(G) = \sum_i J(i, \pi_i)$, with

$$J(i, \pi_i) = \frac{N}{2} \log \frac{|R_{\{i\} \cup \pi_i, \{i\} \cup \pi_i}|}{|R_{\pi_i, \pi_i}||R_{i,i}|} + \frac{d_{\pi_i} d_i}{2} \log N, \qquad (4)$$

where $d_{\pi_i} = \sum_{j \in \pi_i(G)} d_j$. Given the scoring metric $J(G)$, we are faced with an NP-hard optimization problem on the space of directed acyclic graphs [10]. Because the score decomposes as a sum of local scores, local greedy search heuristics are usually exploited. We adopt such heuristics in our simulations, using hillclimbing. It is also possible to use Markov-chain Monte Carlo (MCMC) techniques to sample from the posterior distribution defined by $P(G|D) \propto \exp(-J(G))$ within our framework; this would in principle allow us to output several high-scoring networks.

## 5 Conditional independence tests using KGV

In this section, we indicate how conditional independence tests can be performed using the KGV, and show how these tests can be used to estimate Markov blankets of nodes.

**Likelihood ratio criterion.** In the case of marginal independence, the likelihood ratio criterion is exactly equal to a power of the mutual information (see, e.g, [11] in the case of Gaussian variables). This generalizes easily to conditional independence, where the likelihood ratio criterion to test the conditional independence of $y$ and $z$ given $x$ is equal to $\exp[-N(I(x, y, z) - I(x, y) - I(x, z))]$, where $N$ is the number of samples and the mutual information terms are computed using empirical distributions.

Applied to our Gaussian variables $\phi^G$, we obtain a test statistic based on linear combination of KGV-mutual information terms: $I^K(x, y, z) - I^K(x, y) - I^K(x, z)$. Theoretical threshold values exist for conditional independence tests with Gaussian variables [7], but instead, we prefer to use the value given by the MDL/BIC criterion, i.e., $\frac{1}{2} \frac{\log N}{N} d_y d_z$ (where $d_y$ and $d_z$ are the dimensions of the Gaussians), so that the same decision regarding conditional independence is made in the two approaches (scoring metric or independence tests) [12].

**Markov blankets.** For Gaussian variables, it is well-known that some conditional independencies can be read out from the inverse of the joint covariance matrix [7]. More precisely,

If $y_1, \ldots, y_m$ are $m$ jointly Gaussian random vectors with dimensions $d_i$, and with covariance matrix $\Sigma$ defined in terms of blocks $\Sigma_{ij} = \text{cov}(y_i, y_j)$, then $y_i$ and $y_j$ are independent given all the other variables if and only if the block $(i, j)$ of $K = \Sigma^{-1}$ is equal to zero. Thus in the sample case, we can read out the edges of the undirected model directly from $K$, using the test statistic $l_{ij} = -\frac{1}{2} \log \frac{|K_{jj} - K_{ji} K_{ii}^{-1} K_{ij}|}{|K_{jj}|}$ with the threshold value $\frac{d_i d_j}{2} \frac{\log N}{N}$. Applied to the variables $y_i = \phi_i^G$ and for all pairs of nodes, we can find an undirected graphical model in polynomial time, and thus a set of Markov blankets [4].

We may also be interested in constructing a directed model from the Markov blankets; however, this transformation is not always possible [7]. Consequently, most approaches use heuristics to define a directed model from a set of conditional independencies [4, 13]. Alternatively, as a pruning step in learning a directed graphical model, the Markov blanket can be safely used by only considering directed models whose moral graph is covered by the undirected graph.

## 6 Experiments

We compare the performance of three hillclimbing algorithms for directed graphical models, one using the KGV metric (with $\kappa = 0.01$ and $\sigma = 1$), one using the MDL/BIC metric of [2] and one using the BDe metric of [1] (with equivalent prior sample size $N' = 1$).

When the domain includes continuous variables, we used two discretization strategies; the first one is to use K-means with a given number of clusters, the second one uses the adaptive discretization scheme for the MDL/BIC scoring metric of [14]. Also, to parameterize the local conditional probabilities we used mixture models (mixture of Gaussians, mixture of softmax regressions, mixture of linear regressions), which provide enough flexibility at reasonable cost. These models were fitted using penalized maximum likelihood, and invoking the EM algorithm whenever necessary. The number of mixture components was less than four and determined using the minimum description length (MDL) principle.

When the true generating network is known, we measure the performance of algorithms by the KL divergence to the true distribution; otherwise, we report log-likelihood on held-out test data. We use as a baseline the log-likelihood for the maximum likelihood solution to a model with independent components and multinomial or Gaussian densities as appropriate (i.e., for discrete and continuous variables respectively).

**Toy examples.** We tested all three algorithms on a very simple generative model on $m$ binary nodes, where nodes 1 through $m - 1$ point to node $m$. For each assignment $y$ of the $m - 1$ parents, we set $p(x_m = 1|y)$ by sampling uniformly at random in $[0, 1]$. We also studied a linear Gaussian generative model with the identical topology, with regression weights chosen uniformly at random in $[-1, 1]$. We generated $N = 1000$ samples.

We report average results (over 20 replications) in Figure 1 (left), for $m$ ranging from 1 to 10. We see that on the discrete networks, the performance of all three algorithms is similar, degrading slightly as $m$ increases. On the linear networks, on the other hand, the discretization methods degrade significantly as $m$ increases. The KGV approach is the only approach of the three capable of discovering these simple dependencies in both kinds of networks.

**Discrete networks.** We used three networks commonly used as benchmarks[1], the ALARM network (37 variables), the INSURANCE network (27 variables) and the HAILFINDER network (56 variables). We tested various numbers of samples $N$. We performed 40 replications and report average results in Figure 1 (right). We see that the performance of our metric lies between the (approximate Bayesian) BIC metric and the (full Bayesian) BDe

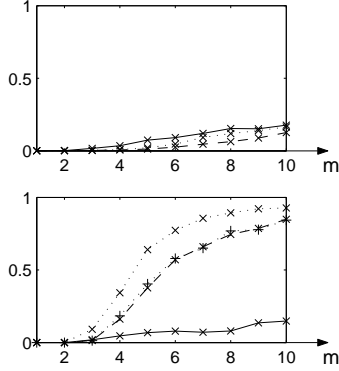

| Network | N ($\times 10^3$) | BIC | BDe | KGV |
|---|---|---|---|---|
| ALARM | 0.5 | 0.85 | 0.47 | 0.66 |
| | 1 | 0.42 | 0.25 | 0.39 |
| | 4 | 0.17 | 0.07 | 0.15 |
| | 16 | 0.04 | 0.02 | 0.06 |
| INSURANCE | 0.5 | 1.84 | 0.92 | 1.53 |
| | 1 | 0.93 | 0.52 | 0.83 |
| | 4 | 0.27 | 0.15 | 0.40 |
| | 16 | 0.05 | 0.04 | 0.19 |
| HAILFINDER | 0.5 | 2.98 | 2.29 | 2.99 |
| | 1 | 1.70 | 1.32 | 1.77 |
| | 4 | 0.63 | 0.48 | 0.63 |
| | 16 | 0.25 | 0.17 | 0.32 |

Figure 1: (Top left) KL divergence vs. size of discrete network $m$: KGV (plain), BDe (dashed), MDL/BIC (dotted). (Bottom left) KL divergence vs. size of linear Gaussian network: KGV (plain), BDe with discretized data (dashed), MDL/BIC with discretized data (dotted x), MDL/BIC with adaptive discretization (dotted +). (Right) KL divergence for discrete network benchmarks.

| Network | N | D | C | d-5 | d-10 | KGV |
|---|---|---|---|---|---|---|
| ABALONE | 4175 | 1 | 8 | 10.68 | 10.53 | **11.16** |
| VEHICLE | 846 | 1 | 18 | 21.92 | 21.12 | **22.71** |
| PIMA | 768 | 1 | 8 | 3.18 | 3.14 | **3.30** |
| AUSTRALIAN | 690 | 9 | 6 | 5.26 | 5.11 | **5.40** |
| BREAST | 683 | 1 | 10 | 15.00 | 15.03 | **15.04** |
| BALANCE | 625 | 1 | 4 | 1.97 | **2.03** | 1.88 |
| HOUSING | 506 | 1 | 13 | **14.71** | 14.25 | 14.16 |
| CARS1 | 392 | 1 | 7 | **6.93** | 6.58 | 6.85 |
| CLEVE | 296 | 8 | 6 | 2.66 | 2.57 | **2.68** |
| HEART | 270 | 9 | 5 | 1.34 | **1.36** | 1.32 |

Table 1: Performance for hybrid networks. $N$ is the number of samples, and $D$ and $C$ are the number of discrete and continuous variables, respectively. The best performance in each row is indicated in bold font.

metric. Thus the performance of the new metric appears to be competitive with standard metrics for discrete data, providing some assurance that even in this case pairwise sufficient statistics in feature space seem to provide a reasonable characterization of Bayesian network structure.

**Hybrid networks.** It is the case of hybrid discrete/continuous networks that is our principal interest—in this case the KGV metric can be applied directly, without discretization of the continuous variables. We investigated performance on several hybrid datasets from the UCI machine learning repository, dividing them into two subsets, 4/5 for training and 1/5 for testing. We also log-transformed all continuous variables that represent rates or counts. We report average results (over 10 replications) in Table 1 for the KGV metric and for the BDe metric—continuous variables are discretized using K-means with 5 clusters (d-5) or 10 clusters (d-10). We see that although the BDe methods perform well in some problems, their performance overall is not as consistent as that of the KGV metric.

## 7 Conclusion

We have presented a general method for learning the structure of graphical models, based on treating variables as Gaussians in a high-dimensional feature space. The method seamlessly integrates discrete and continuous variables in a unified framework, and can provide

improvements in performance when compared to approaches based on discretization of continuous variables.

The method also has appealing computational properties; in particular, the Gaussianity assumption enables us make only a single pass over the data in order to compute the pairwise sufficient statistics. The Gaussianity assumption also provides a direct way to approximate Markov blankets for undirected graphical models, based on the classical link between conditional independence and zeros in the precision matrix.

While the use of the KGV as a scoring metric is inspired by the relationship between the KGV and the mutual information, it must be emphasized that this relationship is a local one, based on an expansion of the mutual information around independence. While our empirical results suggest that the KGV is also an effective surrogate for the mutual information more generally, further theoretical work is needed to provide a deeper understanding of the KGV in models that are far from independence.

Finally, our algorithms have free parameters, in particular the regularization parameter and the width of the Gaussian kernel for continuous variables. Although the performance is empirically robust to the setting of these parameters, learning those parameters from data would not only provide better and more consistent performance, but it would also provide a principled way to learn graphical models with local structure [15].

### Acknowledgments

The simulations were performed using Kevin Murphy's Bayes Net Toolbox for MATLAB. We would like to acknowledge support from NSF grant IIS-9988642, ONR MURI N00014-00-1-0637 and a grant from Intel Corporation.

## Footnotes

[1]Available at http://www.cs.huji.ac.il/labs/compbio/Repository/.

### References

[1] D. Heckerman, D. Geiger, and D. M. Chickering. Learning Bayesian networks: The combination of knowledge and statistical data. *Machine Learning*, 20(3):197–243, 1995.

[2] W. Lam and F. Bacchus. Learning Bayesian belief networks: An approach based on the MDL principle. *Computational Intelligence*, 10(4):269–293, 1994.

[3] D. Geiger and D. Heckerman. Learning Gaussian networks. In *Proc. UAI*, 1994.

[4] J. Pearl. *Causality: Models, Reasoning and Inference*. Cambridge University Press, 2000.

[5] S. Della Pietra, V. J. Della Pietra, and J. D. Lafferty. Inducing features of random fi elds. *IEEE Trans. PAMI*, 19(4):380–393, 1997.

[6] F. R. Bach and M. I. Jordan. Kernel independent component analysis. *Journal of Machine Learning Research*, 3:1–48, 2002.

[7] S. L. Lauritzen. *Graphical Models*. Clarendon Press, 1996.

[8] B. Schölkopf and A. J. Smola. *Learning with Kernels*. MIT Press, 2001.

[9] T. M. Cover and J. A. Thomas. *Elements of Information Theory*. Wiley & Sons, 1991.

[10] D. M. Chickering. Learning Bayesian networks is NP-complete. In *Learning from Data: Artificial Intelligence and Statistics 5*. Springer-Verlag, 1996.

[11] T. W. Anderson. *An Introduction to Multivariate Statistical Analysis*. Wiley & Sons, 1984.

[12] R. G. Cowell. Conditions under which conditional independence and scoring methods lead to identical selection of Bayesian network models. In *Proc. UAI*, 2001.

[13] D. Margaritis and S. Thrun. Bayesian network induction via local neighborhoods. In *Adv. NIPS 12*, 2000.

[14] N. Friedman and M. Goldszmidt. Discretizing continuous attributes while learning Bayesian networks. In *Proc. ICML*, 1996.

[15] N. Friedman and M. Goldszmidt. Learning Bayesian networks with local structure. In *Learning in Graphical Models*. MIT Press, 1998.
